# A Marginalized Particle Gaussian Process Regression

**Yali Wang** and **Brahim Chaib-draa**
Department of Computer Science
Laval University
Quebec, Quebec G1V0A6
{wang,chaib}@damas.ift.ulaval.ca

## Abstract

We present a novel marginalized particle Gaussian process (MPGP) regression, which provides a fast, accurate online Bayesian filtering framework to model the latent function. Using a state space model established by the data construction procedure, our MPGP recursively filters out the estimation of hidden function values by a Gaussian mixture. Meanwhile, it provides a new online method for training hyperparameters with a number of weighted particles. We demonstrate the estimated performance of our MPGP on both simulated and real large data sets. The results show that our MPGP is a robust estimation algorithm with high computational efficiency, which outperforms other state-of-art sparse GP methods.

## 1 Introduction

The Gaussian process (GP) is a popular nonparametric Bayesian method for nonlinear regression. However, the $O(n^3)$ computational load for training the GP model would severely limit its applicability in practice when the number of training points $n$ is larger than a few thousand [1]. A number of attempts have been made to handle it with a small computational load. One typical method is a sparse pseudo-input Gaussian process (SPGP) [2] that uses a pseudo-input data set with $m$ inputs ($m \ll n$) to parameterize the GP predictive distribution to reduce the computational burden. Then a sparse spectrum Gaussian process (SSGP) [3] was proposed to further improve the performance of SPGP while retaining the computational efficiency by using a stationary trigonometric Bayesian model with $m$ basis functions. However, both SPGP and SSGP learn hyperparameters offline by maximizing the marginal likelihood before making the inference. They would take a risk to fall in the local optimum. Another recent model is a Kalman filter Gaussian process (KFGP) [4] which reduces computation load by correlating function values of data subsets at each Kalman filter iteration. But it still causes underfitting or overfitting if the hyperparameters are badly learned offline.

On the contrary, we propose in this paper an online marginalized particle filter to simultaneously learn the hyperprameters and hidden function values. By collecting small data subsets sequentially, we establish a novel state space model which allows us to estimate the marginal posterior distribution (not the marginal likelihood) of hyperparameters online with a number of weighted particles. For each particle, a Kalman filter is applied to estimate the posterior distribution of hidden function values. We will later explain it in details and show its validity via the experiments on large datasets.

## 2 Data Construction

In practice, the whole training data set is usually constructed by gathering small subsets several times. For the $t_{th}$ collection, the training subset $(X_t, \mathbf{y}_t)$ consists of $n_t$ input-output pairs: $\{(\mathbf{x}_t^1, y_t^1), \cdots (\mathbf{x}_t^{n_t}, y_t^{n_t})\}$. Each scalar output $y_t^i$ is generated from a nonlinear function $f(\mathbf{x}_t^i)$ of a d-dimensional input vector $\mathbf{x}_t^i$ with an additive Gaussian noise $N(0, a_0^2)$. All the pairs are separately organized as an input matrix $X_t$ and output vector $\mathbf{y}_t$. For simplicity, the whole training data with

$T$ collections is symbolized as $(X_{1:T}, \mathbf{y}_{1:T})$. The goal refers to a regression issue - estimating the function value of $f(\mathbf{x})$ at $m$ test inputs $X_\star = [\mathbf{x}_\star^1, \cdots \mathbf{x}_\star^m]$ given $(X_{1:T}, \mathbf{y}_{1:T})$.

## 3  Gaussian Process Regression

A Gaussian process (GP) represents a distribution over functions, which is a generalization of the Gaussian distribution to an infinite dimensional function space. Formally, it is a collection of random variables, any finite number of which have a joint Gaussian distribution [1]. Similar to a Gaussian distribution specified by a mean vector and covariance matrix, a GP is fully defined by a mean function $m(\mathbf{x}) = E[f(\mathbf{x})]$ and covariance function $k(\mathbf{x}, \mathbf{x}') = E[(f(\mathbf{x}) - m(\mathbf{x}))(f(\mathbf{x}') - m(\mathbf{x}'))]$. Here we follow the practical choice that $m(\mathbf{x})$ is set to be zero. Moreover, due to the spatial non-stationary phenomena in the real world, we choose $k(\mathbf{x}, \mathbf{x}')$ as $k_{SE}(\mathbf{x}, \mathbf{x}') + k_{NN}(\mathbf{x}, \mathbf{x}')$ where $k_{SE} = a_1^2 exp[-0.5a_2^{-2}(\mathbf{x} - \mathbf{x}')^T(\mathbf{x} - \mathbf{x}')]$ is the stationary squared exponential covariance function, $k_{NN} = a_3^2 sin^{-1}[a_4^{-2}\tilde{\mathbf{x}}^T\tilde{\mathbf{x}}'((1 + a_4^{-2}\tilde{\mathbf{x}}^T\tilde{\mathbf{x}})(1 + a_4^{-2}\tilde{\mathbf{x}}'^T\tilde{\mathbf{x}}'))^{-0.5}]$ is the nonstationary neural network covariance function with the augmented input $\tilde{\mathbf{x}} = [1 \ \mathbf{x}^T]^T$. For simplicity, all the hyperparameters are collected into a vector $\theta = [a_0 \ a_1 \ a_2 \ a_3 \ a_4]^T$.

The regression problem could be solved by the standard GP with the following two steps: First, learning $\theta$ given $(X_{1:T}, \mathbf{y}_{1:T})$. One technique is to draw samples from $p(\theta|X_{1:T}, \mathbf{y}_{1:T})$ using Markov Chain Monte Carlo (MCMC) [5, 6], another popular way is to maximize the log evidence $p(\mathbf{y}_{1:T}|X_{1:T}, \theta)$ via a gradient based optimizer [1]. Second, estimating the distribution of the function value $p(f(X_\star)|X_{1:T}, \mathbf{y}_{1:T}, X_\star, \theta)$. From the perspective of GP, a function $f(\mathbf{x})$ could be loosely considered as an infinitely long vector in which each random variable is the function value at an input $\mathbf{x}$, and any finite set of function values is jointly Gaussian distributed. Hence, the joint distribution $p(\mathbf{y}_{1:T}, f(X_\star)|X_{1:T}, X_\star, \theta)$ is a multivariate Gaussian distribution. Then according to the conditional property of Gaussian distribution, $p(f(X_\star)|X_{1:T}, \mathbf{y}_{1:T}, X_\star, \theta)$ is also Gaussian distributed with the following mean vector $\bar{f}(X_\star)$ and covariance matrix $P(X_\star, X_\star)$ [1, 7]:

$$\bar{f}(X_\star) = K_\theta(X_\star, X_{1:T})[K_\theta(X_{1:T}, X_{1:T}) + a_0^2 I]^{-1}\mathbf{y}_{1:T}$$

$$P(X_\star, X_\star) = K_\theta(X_\star, X_\star) - K_\theta(X_\star, X_{1:T})[K_\theta(X_{1:T}, X_{1:T}) + a_0^2 I]^{-1}K_\theta(X_\star, X_{1:T})^T$$

If there are $n$ training inputs and $m$ test inputs then $K_\theta(X_\star, X_{1:T})$ denotes an $m \times n$ covariance matrix in which each entry is calculated by the covariance function $k(\mathbf{x}, \mathbf{x}')$ with the learned $\theta$. It is similar to construct $K_\theta(X_{1:T}, X_{1:T})$ and $K_\theta(X_\star, X_\star)$.

## 4  Marginalized Particle Gaussian Process Regression

Even though GP is an elegant nonparametric method for Bayesian regression, it is commonly infeasible for large data sets due to an $O(n^3)$ scaling for learning the model. In order to derive a computational tractable GP model which preserves the estimation accuracy, we firstly explore a state space model from the data construction procedure, then propose a marginalized particle filter to estimate the hidden $f(X_\star)$ and $\theta$ in an online Bayesian filtering framework.

### 4.1  State Space Model

The standard state space model (SSM) consists of the state equation and observation equation. The state equation reflects the Markovian evolution of hidden states (the hyperparamters and function values). For the hidden static hyperparameter $\theta$, a popular method in filtering techniques is to add an artificial evolution using kernel smoothing which guarantees the estimation convergence [8, 9, 10]:

$$\theta_t = b\theta_{t-1} + (1 - b)\bar{\theta}_{t-1} + s_{t-1} \tag{1}$$

where $b = (3\delta - 1)/(2\delta)$, $\delta$ is a discount factor which is typically around 0.95-0.99, $\bar{\theta}_{t-1}$ is the Monte Carlo mean of $\theta$ at $t - 1$, and $s_{t-1} \sim N(0, r^2\Sigma_{t-1})$, $r^2 = 1 - b^2$, $\Sigma_{t-1}$ is the Monte Carlo variance matrix of $\theta$ at $t - 1$. For hidden function values, we attempt to explore the relation between the $(t-1)_{th}$ and $t_{th}$ data subset. For simplicity, we denoted $X_t^c = X_t \cup X_\star$ and $f_t^c = f(X_t^c)$. If $f(\mathbf{x}) \sim GP(0, k(\mathbf{x}, \mathbf{x}'))$, then the prior distribution $p(f_t^c, f_{t-1}^c|X_{t-1}^c, X_t^c, \theta_t)$ is jointly Gaussian:

$$p(f_t^c, f_{t-1}^c|X_{t-1}^c, X_t^c, \theta_t) = N(\mathbf{0}, \begin{bmatrix} K_{\theta_t}(X_t^c, X_t^c) & K_{\theta_t}(X_t^c, X_{t-1}^c) \\ K_{\theta_t}(X_t^c, X_{t-1}^c)^T & K_{\theta_t}(X_{t-1}^c, X_{t-1}^c) \end{bmatrix})$$

Then according to the conditional property of Gaussian distribution, we could get

$$p(f_t^c | f_{t-1}^c, X_{t-1}^c, X_t^c, \theta_t) = N(G(\theta_t) f_{t-1}^c, Q(\theta_t)) \tag{2}$$

where

$$G(\theta_t) = K_{\theta_t}(X_t^c, X_{t-1}^c) K_{\theta_t}^{-1}(X_{t-1}^c, X_{t-1}^c) \tag{3}$$

$$Q(\theta_t) = K_{\theta_t}(X_t^c, X_t^c) - K_{\theta_t}(X_t^c, X_{t-1}^c) K_{\theta_t}^{-1}(X_{t-1}^c, X_{t-1}^c) K_{\theta_t}(X_t^c, X_{t-1}^c)^T \tag{4}$$

This conditional density (2) could be transformed into a linear equation of the function value with an additive Gaussian noise $v_t^f \sim N(\mathbf{0}, Q(\theta_t))$:

$$f_t^c = G(\theta_t) f_{t-1}^c + v_t^f \tag{5}$$

Finally, the observation (output) equation could be directly obtained from the $t_{th}$ data collection:

$$\mathbf{y}_t = H_t f_t^c + v_t^y \tag{6}$$

where $H_t = [I_{n_t} \quad \mathbf{0}]$ is an index matrix to make $H_t f_t^c = f(X_t)$ since $\mathbf{y}_t$ is only obtained from the $t_{th}$ training inputs $X_t$. The noise $v_t^y \sim N(0, R(\theta_t))$ is from the section 2 where $R(\theta_t) = a_{0,t}^2 I$. Note that $a_0$ is a fixed unknown hyperparameter. We use the symbol $a_{0,t}$ just because of the consistency with the artificial evolution of $\theta$. To sum up, our SSM is fully specified by (1), (5), (6).

## 4.2 Bayesian Inference by Marginalized Particle Filter

In contrast to the GP regression with a two-step offline inference in section 3, we propose an online filtering framework to simultaneously learn hyperparameters and estimate hidden function values. According to the SSM before, the inference problem refers to compute the posterior distribution $p(f_t^c, \theta_{1:t} | X_{1:t}, X_\star, \mathbf{y}_{1:t})$. One technique is MCMC, but MCMC usually suffers from a long convergence time. Hence we choose another popular technique - particle filter. However, for our SSM, the traditional sampling importance resampling (SIR) particle filter will introduce the unnecessary computational load due to the fact that (5) in the SSM is a linear structure given $\theta_t$. This inspires us to apply a more efficient marginalized particle filter (also called Rao-Blackwellised particle filter) [9, 11, 12, 13] to deal with the estimation problem by combining Kalman filter into particle filter. Using Bayes rule, the posterior could be factorized as

$$p(f_t^c, \theta_{1:t} | X_{1:t}, X_\star, \mathbf{y}_{1:t}) = p(\theta_{1:t} | X_{1:t}, X_\star, \mathbf{y}_{1:t}) p(f_t^c | \theta_{1:t}, X_{1:t}, X_\star, \mathbf{y}_{1:t})$$

$p(\theta_{1:t} | X_{1:t}, X_\star, \mathbf{y}_{1:t})$ refers to a marginal posterior which could be solved by particle filter. After obtaining the estimation of $\theta_{1:t}$, the second term $p(f_t^c | \theta_{1:t}, X_{1:t}, X_\star, \mathbf{y}_{1:t})$ could be computed by Kalman filter since $f_t^c$ is the hidden state in the linear substructure (equation (5)) of SSM.

The detailed inference procedure is as follows: First, $p(\theta_{1:t} | X_{1:t}, X_\star, \mathbf{y}_{1:t})$ should be factorized in a recursive form so that it could be applied into sequential importance sampling framework:

$$p(\theta_{1:t} | X_{1:t}, X_\star, \mathbf{y}_{1:t}) \propto p(\mathbf{y}_t | \mathbf{y}_{1:t-1}, \theta_{1:t}, X_{1:t}, X_\star) p(\theta_t | \theta_{t-1}) p(\theta_{1:t-1} | X_{1:t-1}, X_\star, \mathbf{y}_{1:t-1})$$

At each iteration of the sequential importance sampling, the particles for the hyperparameter vector are drawn from the proposal distribution $p(\theta_t | \theta_{t-1})$ (easily obtained from equation (1)), then the importance weight for each particle at $t$ could be computed according to $p(\mathbf{y}_t | \mathbf{y}_{1:t-1}, \theta_{1:t}, X_{1:t}, X_\star)$. This distribution could be solved analytically:

$$p(\mathbf{y}_t | \mathbf{y}_{1:t-1}, \theta_{1:t}, X_{1:t}, X_\star) = \int p(\mathbf{y}_t, f_t^c | \mathbf{y}_{1:t-1}, \theta_{1:t}, X_{1:t}, X_\star) df_t^c$$

$$= \int p(\mathbf{y}_t | f_t^c, \theta_t, X_t, X_\star) p(f_t^c | \mathbf{y}_{1:t-1}, \theta_{1:t}, X_{1:t}, X_\star) df_t^c$$

$$= \int N(H_t f_t^c, R(\theta_t)) N(f_{t|t-1}^c, P_{t|t-1}^c) df_t^c$$

$$= N(H_t f_{t|t-1}^c, H_t P_{t|t-1}^c H_t^T + R(\theta_t)) \tag{7}$$

where $p(\mathbf{y}_t | f_t^c, \theta_t, X_t, X_\star)$ follows a Gaussian distribution $N(H_t f_t^c, R(\theta_t))$ (refers to equation (6)), $p(f_t^c | \mathbf{y}_{1:t-1}, \theta_{1:t}, X_{1:t}, X_\star) = N(f_{t|t-1}^c, P_{t|t-1}^c)$ is the prediction step of Kalman filter for $f_t^c$ which is also Gaussian distributed with the predictive mean $f_{t|t-1}^c$ and covariance $P_{t|t-1}^c$.

Second, we explain how to compute $p(f_t^c|\theta_{1:t}, X_{1:t}, X_\star, \mathbf{y}_{1:t})$ using the prediction-update Kalman filter. According to the recursive Bayesian filtering, this posterior could be factorized as:

$$p(f_t^c|\theta_{1:t}, X_{1:t}, X_\star, \mathbf{y}_{1:t}) = \frac{p(\mathbf{y}_t|f_t^c, \theta_t, X_t, X_\star)p(f_t^c|\mathbf{y}_{1:t-1}, \theta_{1:t}, X_{1:t}, X_\star)}{p(\mathbf{y}_t|\mathbf{y}_{1:t-1}, \theta_{1:t}, X_{1:t}, X_\star)} \tag{8}$$

In the prediction step, the goal is to compute $p(f_t^c|\mathbf{y}_{1:t-1}, \theta_{1:t}, X_{1:t}, X_\star)$ which is an integral:

$$
\begin{aligned}
p(f_t^c|\mathbf{y}_{1:t-1}, \theta_{1:t}, X_{1:t}, X_\star) &= \int p(f_t^c, f_{t-1}^c|\mathbf{y}_{1:t-1}, \theta_{1:t}, X_{1:t}, X_\star)df_{t-1}^c \\
&= \int p(f_t^c|f_{t-1}^c, \theta_t, X_{t-1:t}, X_\star)p(f_{t-1}^c|\mathbf{y}_{1:t-1}, \theta_{1:t-1}, X_{1:t-1}, X_\star)df_{t-1}^c \\
&= \int N(G(\theta_t)f_{t-1}^c, Q(\theta_t))N(f_{t-1|t-1}^c, P_{t-1|t-1}^c)df_{t-1}^c \\
&= N(G(\theta_t)f_{t-1|t-1}^c, \ G(\theta_t)P_{t-1|t-1}^c G(\theta_t)^T + Q(\theta_t)) \tag{9}
\end{aligned}
$$

where $p(f_t^c|f_{t-1}^c, \theta_t, X_{t-1:t}, X_\star)$ is directly from (2), and $p(f_{t-1}^c|\mathbf{y}_{1:t-1}, \theta_{1:t-1}, X_{1:t-1}, X_\star) = N(f_{t-1|t-1}^c, P_{t-1|t-1}^c)$ is the posterior estimation for $f_{t-1}^c$. Since $p(f_t^c|\mathbf{y}_{1:t}, \theta_{1:t}, X_{1:t}, X_\star)$ could also be expressed as $N(f_{t|t-1}^c, P_{t|t-1}^c)$, then the prediction step is summarized as:

$$f_{t|t-1}^c = G(\theta_t)f_{t-1|t-1}^c, \quad P_{t|t-1}^c = G(\theta_t)P_{t-1|t-1}^c G(\theta_t)^T + Q(\theta_t) \tag{10}$$

In the update step, the current observation density $p(\mathbf{y}_t|f_t^c, \theta_t, X_t, X_\star) = N(H_t f_t^c, R(\theta_t))$ is used to correct the prediction. Putting (7) and (9) into (8), $p(f_t^c|\theta_{1:t}, X_{1:t}, X_\star, \mathbf{y}_{1:t}) = N(f_{t|t}^c, P_{t|t}^c)$ is actually Gaussian distributed with the Kalman Gain $\Gamma_t$ where:

$$\Gamma_t = P_{t|t-1}^c H_t^T (H_t P_{t|t-1}^c H_t^T + R(\theta_t))^{-1} \tag{11}$$

$$f_{t|t}^c = f_{t|t-1}^c + \Gamma_t(\mathbf{y}_t - H_t f_{t|t-1}^c), \quad P_{t|t}^c = P_{t|t-1}^c - \Gamma_t H_t P_{t|t-1}^c \tag{12}$$

Finally, the whole algorithm ($t = 1, 2, 3, ....$) is summarized as follows:

- For $i = 1, 2, ...N$
    - Drawing $\theta_t^i \sim p(\theta_t|\tilde{\theta}_{t-1}^i)$ according to (1)
    - Using $\theta_t^i$ to specify $k(\mathbf{x}, \mathbf{x}')$ in GP to construct $G(\theta_t^i)$, $Q(\theta_t^i)$, $R(\theta_t^i)$ in (3-4) and (6)
    - Kalman Predict: Using $\tilde{f}_{t-1|t-1}^{c,i}$, $\tilde{P}_{t-1|t-1}^{c,i}$ into (10) to compute $f_{t|t-1}^{c,i}$, $P_{t|t-1}^{c,i}$
    - Kalman Update: Using $f_{t|t-1}^{c,i}$ and $P_{t|t-1}^{c,i}$ into (11) and (12) to compute $f_{t|t}^{c,i}$ and $P_{t|t}^{c,i}$
    - Putting $f_{t|t-1}^{c,i}$, $P_{t|t-1}^{c,i}$, $R(\theta_t^i)$ into (7) to compute the importance weight $\bar{w}_t^i$
- Normalizing the weight: $w_t^i = \bar{w}_t^i / (\sum_{i=1}^N \bar{w}_t^i) \ (i = 1, ...N)$
- Hyperparameter and Hidden function value estimation:
  $\hat{\theta}_t = \sum_{i=1}^N w_t^i \theta_t^i, \quad \hat{f}_{t|t}^c = \sum_{i=1}^N w_t^i f_{t|t}^{c,i} \Rightarrow \hat{f}_{t|t}^\star = H_t^\star \hat{f}_{t|t}^c$
  $\hat{P}_{t|t}^c = \sum_{i=1}^N w_t^i(P_{t|t}^{c,i} + (f_{t|t}^{c,i} - \hat{f}_{t|t}^c)(f_{t|t}^{c,i} - \hat{f}_{t|t}^c)^T) \Rightarrow \hat{P}_{t|t}^\star = H_t^\star \hat{P}_{t|t}^c (H_t^\star)^T$
  where $H_t^\star = [\mathbf{0} \ I_m]$ is an index matrix to get the function value estimation at $X_\star$
- Resampling: For $i = 1, ...N$, resample $\theta_t^i, f_{t|t}^{c,i}, P_{t|t}^{c,i}$ with respect to the importance weight $w_t^i$ to obtain $\tilde{\theta}_t^i, \tilde{f}_{t|t}^{c,i}, \tilde{P}_{t|t}^{c,i}$ for the next step

At each iteration, our marginalized particle Gaussian process (MPGP) uses a small training subset to estimate $f(X_\star)$ by Kalman filters, and learn hyperparameters online by weighted particles. The computational cost of the marginalized particle filter is governed by O($NTS^3$) [10] where $N$ is the number of particles, $T$ is the number of data collections, $S$ is the size of each collection. This could largely reduce the computational load. Moreover, the MPGP propagates the previous estimation to improve the current accuracy in the recursive filtering framework. From the algorithm above, we also find that $f(X_\star)$ is estimated as a Gaussian mixture at each iteration since each hyperparameter particle accompanies with a Kalman filter for $f(X_\star)$. Hence the MPGP could accelerate the

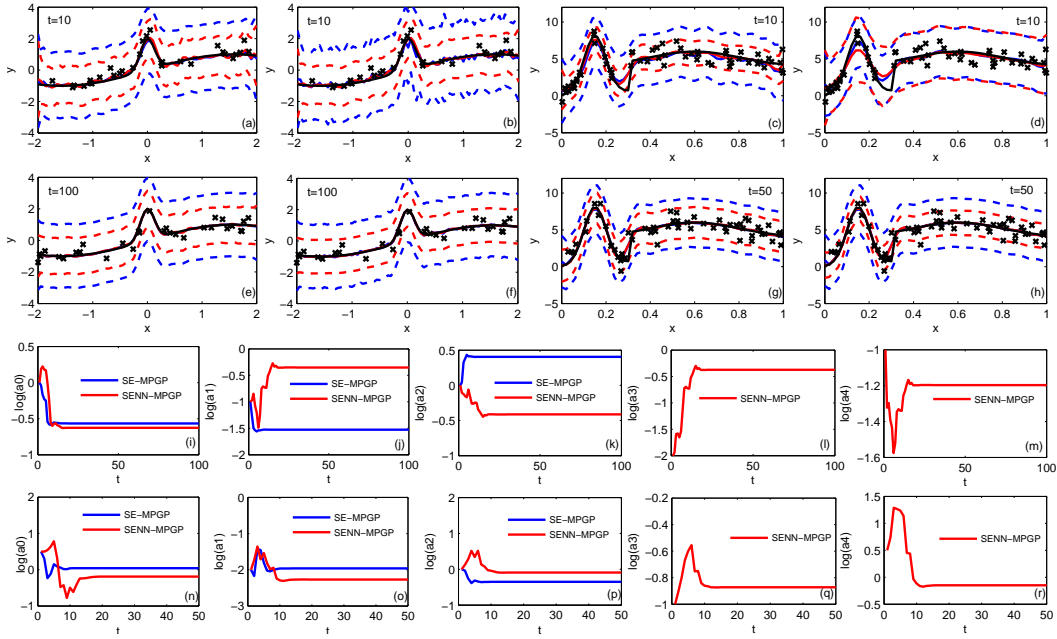

Figure 1: Estimation result comparison. (a-b) show the estimation for $f_1$ at $t = 10$ by SE-KFGP (blue line with blue dashed interval in (a)), SE-MPGP (red line with red dashed interval in (a)), SENN-KFGP (blue line with blue dashed interval in (b)), SENN-MPGP (red line with red dashed interval in (b)). The black crosses are the training outputs at $t = 10$, the black line is the true $f(X_\star)$. The denotation of (c-d),(e-f),(g-h) is same as (a-b) except that (c-d) are for $f_2$ at $t = 10$, (e-f) are for $f_1$ at $t = 100$, (g-h) are for $f_2$ at $t = 50$. (i-m), (n-r) are the estimation of the log hyperparameters ($log(a_0)$ to $log(a_4)$) for $f_1$, $f_2$ over time.

computational speed, while preserving the accuracy. Additionally, it is worth to mention that the Kalman filter GP (KFGP) [4] is a special case of our MPGP since the KFGP firstly trains the hyperparamter vector offline and uses it to specify the SSM, then estimates $p(f_t^c|\theta_{1:t}, X_{1:t}, X_\star, \mathbf{y}_{1:t})$ by Kalman filter. But the offline learning procedure in KFGP will either take a long time using a large extra training data or fall into an unsatisfactory local optimum using a small extra training data. In our MPGP, the local optimum could be used as the initial setting of hyperparameters, then the underlying $\theta$ could be learned online by the marginalized particle filter to improve the performance. Finally, to avoid confusion, we should clarify the difference between our MPGP and the GP modeled Bayesian filters [14, 15]. The goal of GP modeled Bayesian filters is to use GP modeling for Bayesian filtering, on the contrary, our MPGP is to use Bayesian filtering for GP modeling.

## 5   Experiments

**Two Synthetic Datasets**:  The proposed MPGP is firstly evaluated on two simulated one-dimensional datasets. One is a function with a sharp peak which is spatially inhomogeneously smooth [16]: $f_1(x) = \sin(x) + 2\exp(-30x^2)$. For $f_1(x)$, we gather the training data with 100 collections. For each collection, we randomly select 30 inputs from [-2, 2], then calculate their outputs by adding a Gaussian noise $N(0, 0.3^2)$ to their function values. The test input is from -2 to 2 with 0.05 interval. The other function is with a discontinuity [17]: if $0 \le x \le 0.3$, $f_2(x) = \mathcal{N}(x; 0.6, 0.2^2) + \mathcal{N}(x; 0.15, 0.05^2)$, if $0.3 < x \le 1$, $f_2(x) = \mathcal{N}(x; 0.6, 0.2^2) + \mathcal{N}(x; 0.15, 0.05^2) + 4$. For $f_2(x)$, we gather the training data with 50 collections. For each collection, we randomly select 60 inputs from [0, 1], then calculate their outputs by adding a Gaussian noise $N(0, 0.8^2)$ to their function values. The test input is from 0 to 1 with 0.02 interval.

The first experiment aims to evaluate the estimation performance in comparison of KFGP in [4]. We denote SE-KFGP, SENN-KFGP as KFGP with the covariance function $k_{SE}$, KFGP with the covariance function $k_{SE} + k_{NN}$. Similarly, SE-MPGP and SENN-MPGP are MPGP with $k_{SE}$,

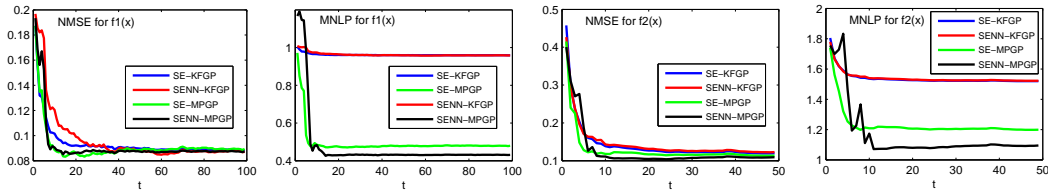

Figure 2: The NMSE and MNLP of KFGP and MPGP for $f_1$, $f_2$ over time.

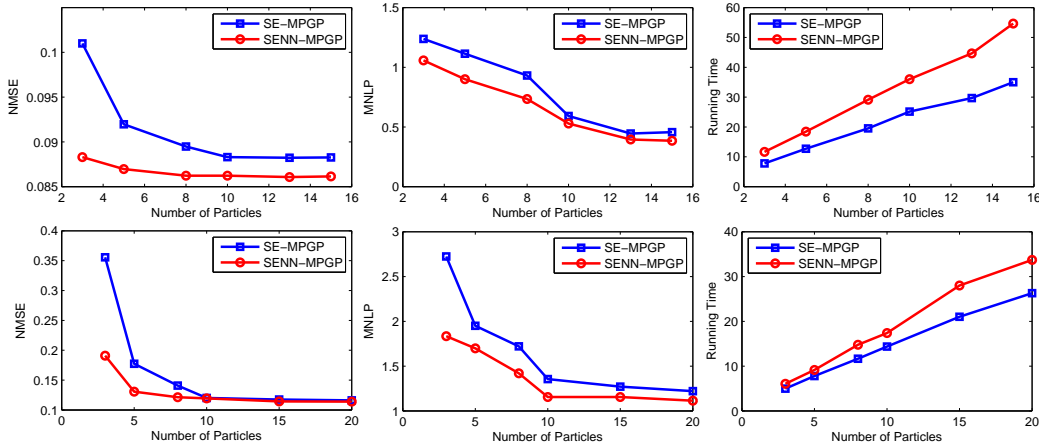

Figure 3: The NMSE and MNLP of MPGP as a function of the number of particles. The first row is for $f_1$, the second row is for $f_2$.

MPGP with $k_{SE} + k_{NN}$. The number of particles in MPGP is set to 10. The evaluation criterion is the test Normalized Mean Square Error (NMSE) and the test Mean Negative Log Probability (MNLP) as suggested in [3]. First, it is shown in Figure 1 that the estimation performance for both KFGP and MPGP is getting better and attempts to convergence over time (refers to (a-h)) since the previous estimation would be incorporated into the current estimation in the recursive Bayesian filtering. Second, for both $f_1$ and $f_2$, the estimation of MPGP is better than KFGP via the NMSE and MNLP comparison in Figure 2. The KFGP uses offline learned hyperparameters all the time. On the contrary, MPGP initializes hyperparameters using the ones by KFGP, then online learns the true hyperparameters (refers to (i-r) in Figure 1). Hence the MNLP of MPGP is much lower than KFGP. Finally, if we only focus on our MPGP, then we could find SENN-MPGP is better than SE-MPGP since SENN-MPGP takes the spatial nonstationary phenomenon into account.

The second experiment aims to illustrate the average performance of SE-MPGP and SENN-MPGP when the number of particles increases. For each number of particles, we run the SE-MPGP and SENN-MPGP 5 times and compute the average NMSE and MNLP. From Figure 3, we find: First, with increasing the number of particles, the NMSE and MNLP of SE-MPGP and SENN-MPGP would decrease at the beginning and become convergence while the running time increases over time. The reason is that the estimation accuracy and computational load of particle filters will increase when the number of particles increases. Second, the average performance of SENN-MPGP is better than SE-MPGP since it captures the spatial nonstationarity, but SENN-MPGP needs more running time since the size of the hyperparameter vector to be inferred will increase.

The third experiment aims to compare our MPGP with the benchmarks. The state-of-art sparse GP methods we choose are: sparse pseudo-input Gaussian process (SPGP) [2] and sparse spectrum Gaussian process (SSGP) [3]. Moreover, we also want to examine the robustness of our MPGP since we should clarify whether the good estimation of our MPGP heavily depends on the order of training data collection. Hence, we randomly interrupt the order of training subsets we used before, then implement SPGP with 5 pseudo inputs (5-SPGP), SSGP with 10 basis functions (10-SSGP), SE-MPGP with 5 particles (5-SE-MPGP), SENN-MPGP with 5 particles (5-SENN-MPGP).

Table 1: Benchmarks Comparison for Synthetic Datasets. The NMSEi, MNLPi, RTimei represent the NMSE, MNLP and running time for the function $f_i$ $(i = 1, 2)$

| Method | NMSE1 | MNLP1 | RTime1 | NMSE2 | MNLP2 | RTime2 |
|---|---|---|---|---|---|---|
| 5-SPGP | 0.2243 | 0.5409 | 28.6418s | 0.5445 | 1.5950 | 30.3578s |
| 10-SSGP | 0.0887 | 0.1606 | 18.8605s | 0.1144 | 1.1208 | 10.2025s |
| 5-SE-MPGP | 0.0880 | 1.6318 | 12.5737s | 0.1687 | 1.3524 | 12.4801s |
| 5-SENN-MPGP | 0.0881 | 0.1820 | 18.7513s | 0.1289 | 1.1782 | 11.5909s |

Table 2: Benchmarks Comparison. Data1 is the temperature dataset. Data2 is the pendulum dataset.

| Data1 | NMSE | MNLP | RTime | Data2 | NMSE | MNLP | RTime |
|---|---|---|---|---|---|---|---|
| 5-SPGP | 0.48 | 1.62 | 181.3s | 10-SPGP | 0.61 | 1.98 | 16.54s |
| 10-SSGP | 0.27 | 1.33 | 97.16s | 10-SSGP | 1.04 | 10.85 | 23.59s |
| 5-SE-MPGP | 0.11 | 1.05 | 50.99s | 20-SE-MPGP | 0.63 | 2.20 | 7.04s |
| 5-SENN-MPGP | 0.10 | 1.16 | 59.25s | 20-SENN-MPGP | 0.58 | 2.12 | 8.60s |

In Table 1, our 5-SE-MPGP mainly outperforms SPGP except that its MNLP1 is worse than the one of SPGP. The reason is the synthetic functions are nonstationary but SE-MPGP uses a stationary SE kernel. Hence we perform 5-SENN-MPGP with a nonstationary kernel to show that our MPGP is competitive with SSGP, and much better with shorter running time than SPGP.

**Global Surface Temperature Dataset**: We present here a preliminary analysis of the Global Surface Temperature Dataset in January 2011 (http://data.giss.nasa.gov/gistemp/). We first gather the training data with 100 collections. For each collection, we randomly select 90 data points where the input vector is the longitude and latitude location, the output is the temperature ($^oC$). There are two test data sets: the first one is a grid test input set (Longitude: -180:40:180, Latitude: -90:20:90) that is used to show the estimated surface temperature. The second test input set (100 points) is randomly selected from the data website after obtaining all the training data.

The first experiment aims to show the predicted surface temperature at the grid test inputs. We set the number of particles in the SE-MPGP and SENN-MPGP as 20. From Figure 4, the KFGP methods stuck in the local optimum: SE-KFGP seems underfitting since it does not model the cold region around the location (100, 50), SENN-KFGP seems overfitting since it unexpectedly models the cold region around (-100, -50). On the contrary, SE-MPGP and SENN-MPGP suitably fit the data set via the hyperparameter online learning.

The second experiment is to evaluate the estimation error of our MPGP using the second test data. We firstly run all the methods to compute the NMSE and MNLP over iteration. From the first row of Figure 5, the NMSE and MNLP of MPGP are lower than KFGP. Moreover, SENN-MPGP is much lower than SE-MPGP, which shows that SENN-MPGP successfully models the spatial nonstationarity of the temperature data. Then we change the number of particles. For each number, we run SE-MPGP, SENN-MPGP 3 times to evaluate the average NMSE, MNLP and running time. It shows that SENN-MPGP fits the data better than SE-MPGP but the trade-off is the longer running time.

The third experiment is to compare our MPGP with the benchmarks. All the denotations are same as the third experiment of the simulated data. We also randomly interrupt the order of training subsets for the robustness consideration. From Table 2, the comparison results show that our MPGP uses a shorter running time with a better estimation performance than SPGP and SSGP.

**Pendulum Dataset**: This is a small data set which contains 315 training points. In [3], it is mentioned that SSGP model seems to be overfitting for this data due to the gradient ascent optimization. We are interested in whether our method can successfully capture the nonlinear property of this pendulum data. We firstly collect the training data 9 times, and 35 training data for each collection. Then, 100 test points are randomly selected for evaluating the performance. From Table 2, our SENN-MPGP obtains the estimation with the fastest speed and the smallest NMSE among all the methods, and the MNLP is competitive to SPGP.

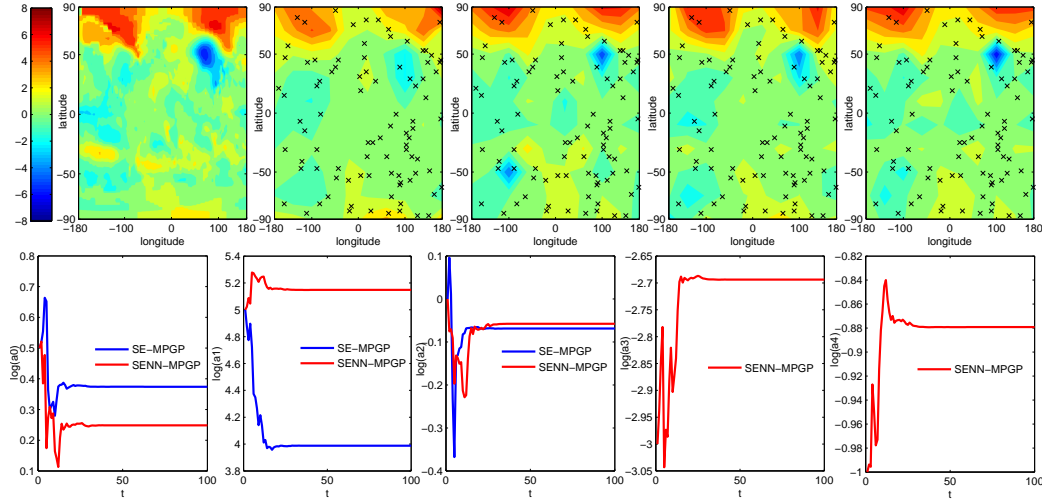

Figure 4: The temperature estimation at $t = 100$. The first row (from left to right): the temperature value bar, the full training observation plot, the grid test output estimation by SE-KFGP, SENN-KFGP, SE-MPGP, SENN-MPGP. The black crosses are the observations at $t = 100$. The second row (from left to right) is the estimation of log hyperparameters ($log(a_0)$ to $log(a_4)$).

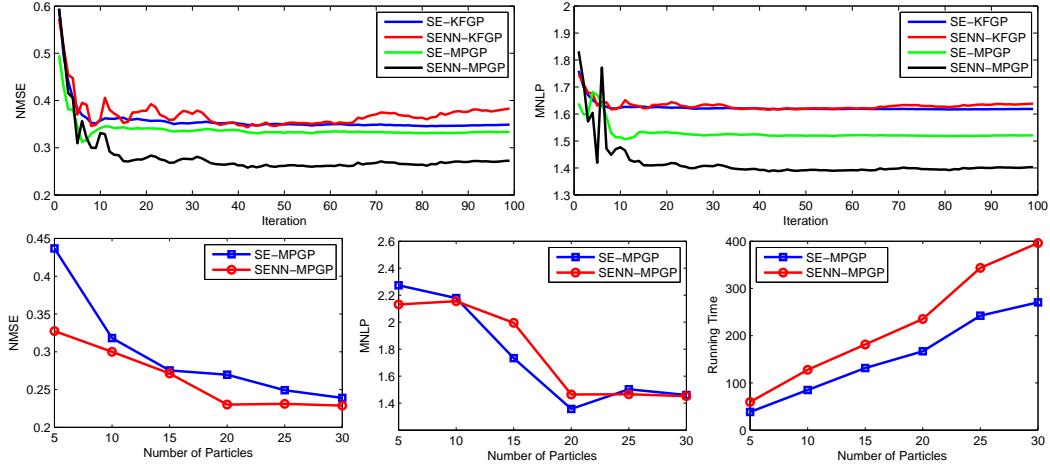

Figure 5: The NMSE and MNLP evaluation. The first row: the NMSE and MNLP over iteration. The second row: the average NMSE, MNLP, Running time as a function of the number of particles.

## 6  Conclusion

We have proposed a novel Bayesian filtering framework for GP regression, which is a fast and accurate online method. Our MPGP framework does not only estimate the function value successfully, but it also provides a new technique for learning the unknown static hyperparameters by online estimating the marginal posterior of hyperparameters. The small training set at each iteration would largely reduce the computation load while the estimation performance is improved over iteration due to the fact that recursive filtering would propagate the previous estimation to enhance the current estimation. In comparison with other benchmarks, we have shown that our MPGP could provide a robust estimation with a competitively computational speed. In the future, it would be interesting to explore the time-varying function estimation with our MPGP.

# References

[1] C. E. Rasmussen, C. K. I. Williams, Gaussian Process for Machine learning, MIT Press, Cambridge, MA, 2006.

[2] E. Snelson, Z. Ghahramani, Sparse gaussian processes using pseudo-inputs, in: NIPS, 2006, pp. 1257–1264.

[3] M. L.-Gredilla, J. Q.-Candela, C. E. Rasmussen, A. R. F.-Vidal, Sparse spectrum gaussian process regression, Journal of Machine Learning Research 11 (2010) 1865–1881.

[4] S. Reece, S. Roberts, An introduction to gaussian processes for the kalman filter expert, in: FUSION, 2010.

[5] R. M. Neal, Monte carlo implementation of gaussian process models for bayesian regression and classification, Tech. rep., Department of Statistics, University of Toronto (1997).

[6] D. J. C. MacKay, Introduction to gaussian processes, in: Neural Networks and Machine Learning, 1998, pp. 133–165.

[7] M. P. Deisenroth, Efficient reinforcement learning using gaussian processes, Ph.D. thesis, Karlsruhe Institute of Technology (2010).

[8] J. Liu, M. West, Combined parameter and state estimation in simulation-based filtering, in: Sequential Monte Carlo Methods in Practice, 2001, pp. 197–223.

[9] P. Li, R. Goodall, V. Kadirkamanathan, Estimation of parameters in a linear state space model using a Rao-Blackwellised particle filter, IEE Proceedings on Control Theory and Applications 151 (2004) 727–738.

[10] N. Kantas, A. Doucet, S. S. Singh, J. M. Maciejowski, An overview of squential Monte Carlo methods for parameter estimation in general state space models, in: 15 th IFAC Symposium on System Identification, 2009.

[11] A. Doucet, N. de Freitas, K. Murphy, S. Russell, Rao-Blackwellised particle filtering for dynamic Bayesian networks, in: UAI, 2000, pp. 176–183.

[12] N. de Freitas, Rao-Blackwellised particle filtering for fault diagnosis, in: IEEE Aerospace Conference Proceedings, 2002, pp. 1767–1772.

[13] T. Schön, F. Gustafsson, P.-J. Nordlund, Marginalized particle filters for mixed linear/nonlinear state-space models, IEEE Transactions on Signal Processing 53 (2005) 2279 – 2289.

[14] J. Ko, D. Fox, Gp-bayesfilters: Bayesian filtering using gaussian process prediction and observation models, in: IROS, 2008, pp. 3471–3476.

[15] M. P. Deisenroth, R. Turner, M. F. Huber, U. D. Hanebeck, C. E. Rasmussen, Robust filtering and smoothing with gaussian processes, IEEE Transactions on Automatic Control.

[16] I. DiMatteo, C. R. Genovese, R. E. Kass, Bayesian Curve Fitting with Free-Knot Splines, Biometrika 88 (2001) 1055–1071.

[17] S. A. Wood, Bayesian mixture of splines for spatially adaptive nonparametric regression, Biometrika 89 (2002) 513–528.

